# Structure learning in human causal induction

**Joshua B. Tenenbaum & Thomas L. Griffiths**
Department of Psychology
Stanford University, Stanford, CA 94305
{jbt,gruffydd}@psych.stanford.edu

## Abstract

We use graphical models to explore the question of how people learn simple causal relationships from data. The two leading psychological theories can both be seen as estimating the parameters of a fixed graph. We argue that a complete account of causal induction should also consider how people learn the underlying causal graph structure, and we propose to model this inductive process as a Bayesian inference. Our argument is supported through the discussion of three data sets.

## 1 Introduction

Causality plays a central role in human mental life. Our behavior depends upon our understanding of the causal structure of our environment, and we are remarkably good at inferring causation from mere observation. Constructing formal models of causal induction is currently a major focus of attention in computer science [7], psychology [3,6], and philosophy [5]. This paper attempts to connect these literatures, by framing the debate between two major psychological theories in the computational language of graphical models. We show that existing theories equate human causal induction with maximum likelihood parameter estimation on a fixed graphical structure, and we argue that to fully account for human behavioral data, we must also postulate that people make Bayesian inferences about the underlying causal graph structure itself.

Psychological models of causal induction address the question of how people learn associations between causes and effects, such as $P(C \rightarrow E)$, the probability that some event $C$ causes outcome $E$. This question might seem trivial at first; why isn't $P(C \rightarrow E)$ simply $P(e^+|c^+)$, the conditional probability that $E$ occurs ($E = e^+$ as opposed to $e^-$) given that $C$ occurs? But consider the following scenarios. Three case studies have been done to evaluate the probability that certain chemicals, when injected into rats, cause certain genes to be expressed. In case 1, levels of gene 1 were measured in 100 rats injected with chemical 1, as well as in 100 uninjected rats; cases 2 and 3 were conducted likewise but with different chemicals and genes. In case 1, 40 out of 100 injected rats were found to have expressed the gene, while 0 out of 100 uninjected rats expressed the gene. We will denote these results as $\{40/100, 0/100\}$. Case 2 produced the results $\{7/100, 0/100\}$, while case 3 yielded $\{53/100, 46/100\}$. For each case, we would like to know the probability that the chemical causes the gene to be expressed, $P(C \rightarrow E)$, where $C$ denotes the chemical and $E$ denotes gene expression.

People typically rate $P(C{\rightarrow}E)$ highest for case 1, followed by case 2 and then case 3. In an experiment described below, these cases received mean ratings (on a 0-20 scale) of $14.9{\pm}.8$, $8.6{\pm}.9$, and $4.9{\pm}.7$, respectively. Clearly $P(C{\rightarrow}E) \neq P(e^+|c^+)$, because case 3 has the highest value of $P(e^+|c^+)$ but receives the lowest rating for $P(C{\rightarrow}E)$.

The two leading psychological models of causal induction elaborate upon this basis in attempting to specify $P(C{\rightarrow}E)$. The $\Delta P$ model [6] claims that people estimate $P(C{\rightarrow}E)$ according to

$$\Delta P = P(e^+|c^+) - P(e^+|c^-). \tag{1}$$

(We restrict our attention here to facilitatory causes, in which case $\Delta P$ is always between 0 and 1.) Equation 1 captures the intuition that $C$ is perceived to cause $E$ to the extent that $C$'s occurence increases the likelihood of observing $E$. Recently, Cheng [3] has identified several shortcomings of $\Delta P$ and proposed that $P(C{\rightarrow}E)$ instead corresponds to *causal power*, the probability that $C$ produces $E$ in the absence of all other causes. Formally, the power model can be expressed as:

$$\text{power} = \frac{\Delta P}{1 - P(e^+|c^-)}. \tag{2}$$

There are a variety of normative arguments in favor of either of these models [3,7]. Empirically, however, neither model is fully adequate to explain human causal induction. We will present ample evidence for this claim below, but for now, the basic problem can be illustrated with the three scenarios above. While people rate $P(C{\rightarrow}E)$ higher for case 2, {7/100,0/100}, than for case 3, {53/100,46/100}, $\Delta P$ rates them equally and the power model ranks case 3 over case 2. To understand this discrepancy, we have to distinguish between two possible senses of $P(C{\rightarrow}E)$: "the probability that C causes E (on any given trial when C is present)" versus "the probability that C is a cause of E (in general, as opposed to being causally independent of E)". Our claim is that the $\Delta P$ and power models concern only the former sense, while people's intuitions about $P(C{\rightarrow}E)$ are often concerned with the latter. In our example, while the effect of $C$ on any given trial in case 3 may be equal to (according to $\Delta P$) or stronger than (according to power) its effect in case 2, the general pattern of results seems more likely in case 2 than in case 3 to be due to a genuine causal influence, as opposed to a spurious correlation between random samples of two independent variables. In the following section, we formalize this distinction in terms of parameter estimation versus structure learning on a graphical model. Section 3 then compares two variants of our structure learning model with the parameter estimation models ($\Delta P$ and power) in light of data from three experiments on human causal induction.

## 2 Graphical models of causal induction

The language of causal graphical models provides a useful framework for thinking about people's causal intuitions [5,7]. All the induction models we consider here can be viewed as computations on a simple directed graph ($\text{Graph}_1$ in Figure 1). The effect node $E$ is the child of two binary-valued parent nodes: $C$, the putative cause, and $B$, a constant background. Let $X = \langle C_1, E_1 \rangle, \ldots, \langle C_N, E_N \rangle$ denote a sequence of $N$ trials in which $C$ and $E$ are each observed to be present or absent; $B$ is assumed to be present on all trials. (To keep notation concise in this section, we use 1 or 0 in addition to $^+$ or $^-$ to denote presence or absence of an event, e.g. $c_i = 1$ if the cause is present on the $i$th trial.) Each parent node is associated with a parameter, $w_B$ or $w_C$, that defines the strength of its effect on $E$. In the

$\Delta P$ model, the probability of $E$ occuring is a linear function of $C$:

$$Q(e^+|c; w_B, w_C) = w_B + w_C \cdot c. \tag{3}$$

(We use $Q$ to denote model probabilities and $P$ for empirical probabilities in the sample $X$.) In the causal power model, as first shown by Glymour [5], $E$ is a noisy-OR gate:

$$Q(e^+|c; w_B, w_C) = 1 - (1 - w_B)(1 - w_C)^c. \tag{4}$$

## 2.1 Parameter inferences: $\Delta P$ and Causal Power

In this framework, both the $\Delta P$ and power model's predictions for $P(C{\rightarrow}E)$ can be seen as maximum likelihood estimates of the causal strength parameter $w_C$ in $\mathrm{Graph}_1$, but under different parameterizations. For either model, the loglikelihood of the data is given by

$$\mathcal{L}(X|w_B, w_C) = \sum_{i=1}^{N} \log \left[ (Q(e_i|c_i))^{e_i} (1 - Q(e_i|c_i))^{1-e_i} \right] \tag{5}$$

$$= \sum_{i=1}^{N} e_i \log Q(e_i|c_i) + (1 - e_i) \log(1 - Q(e_i|c_i)), \tag{6}$$

where we have suppressed the dependence of $Q(e_i|c_i)$ on $w_B, w_C$. Breaking this sum into four parts, one for each possible combination of $\{e^+, e^-\}$ and $\{c^+, c^-\}$ that could be observed, $\mathcal{L}(X|w_B, w_C)$ can be written as

$$N\, P(c^+)\, [P(e^+|c^+) \log Q(e^+|c^+) + (1 - P(e^+|c^+)) \log(1 - Q(e^+|c^+))] \tag{7}$$
$$+\quad N\, P(c^-)\, [P(e^+|c^-) \log Q(e^+|c^-) + (1 - P(e^+|c^-)) \log(1 - Q(e^+|c^-))]$$

By the Information inequality [4], Equation 7 is maximized whenever $w_B$ and $w_C$ can be chosen to make the model probabilities equal to the empirical probabilites:

$$Q(e^+|c^+; w_B, w_C) = P(e^+|c^+), \tag{8}$$
$$Q(e^+|c^-; w_B, w_C) = P(e^+|c^-). \tag{9}$$

To show that the $\Delta P$ model's predictions for $P(C{\rightarrow}E)$ correspond to maximum likelihood estimates of $w_C$ under a linear parameterization of $\mathrm{Graph}_1$, we identify $w_C$ in Equation 3 with $\Delta P$ (Equation 1), and $w_B$ with $P(e^+|c^-)$. Equation 3 then reduces to $P(e^+|c^+)$ for the case $c = c^+$ (i.e., $c = 1$) and to $P(e^+|c^-)$ for the case $c = c^-$ (i.e., $c = 0$), thus satisfying the sufficient conditions in Equations 8-9 for $w_B$ and $w_C$ to be maximum likelihood estimates. To show that the causal power model's predictions for $P(C{\rightarrow}E)$ correspond to maximum likelihood estimates of $w_C$ under a noisy-OR parameterization, we follow the analogous procedure: identify $w_C$ in Equation 4 with power (Equation 2), and $w_B$ with $P(e^+|c^-)$. Then Equation 4 reduces to $P(e^+|c^+)$ for $c = c^+$ and to $P(e^+|c^-)$ for $c = c^-$, again satisfying the conditions for $w_B$ and $w_C$ to be maximum likelihood estimates.

## 2.2 Structural inferences: Causal Support and $\chi^2$

The central claim of this paper is that people's judgments of $P(C{\rightarrow}E)$ reflect something other than estimates of causal strength parameters – the quantities that we have just shown to be computed by $\Delta P$ and the power model. Rather, people's judgments may correspond to inferences about the underlying causal structure, such as the probability that $C$ is a direct

cause of $E$. In terms of the graphical model in Figure 1, human causal induction may be focused on trying to distinguish between $\mathrm{Graph}_1$, in which $C$ is a parent of $E$, and the "null hypothesis" of $\mathrm{Graph}_0$, in which $C$ is not.

This structural inference can be formalized as a Bayesian decision. Let $h_C$ be a binary variable indicating whether or not the link $C \rightarrow E$ exists in the true causal model responsible for generating our observations. We will assume a noisy-OR gate, and thus our model is closely related to causal power. However, we propose to model human estimates of $P(C \rightarrow E)$ as *causal support*, the log posterior odds in favor of $\mathrm{Graph}_1$ ($h_C = 1$) over $\mathrm{Graph}_0$ ($h_C = 0$):

$$\text{support} = \log \frac{P(h_C = 1 | X)}{P(h_C = 0 | X)}. \tag{10}$$

Via Bayes' rule, we can express $P(h_C = 1 | X)$ in terms of the marginal likelihood or *evidence*, $P(X | h_C = 1)$, and the prior probability that $C$ is a cause of $E$, $P(h_C = 1)$:

$$P(h_C = 1 | X) \propto P(X | h_C = 1) P(h_C = 1). \tag{11}$$

For now, we take $P(h_C = 1) = P(h_C = 0) = 1/2$. Computing the evidence requires integrating the likelihood $P(X | w_B, w_C)$ over all possible values of the strength parameters:

$$P(X | h_C = 1) = \int_0^1 \int_0^1 P(X | w_B, w_C) \, p(w_B, w_C | h_C = 1) \, dw_B \, dw_C. \tag{12}$$

We take $p(w_B, w_C | h_C = 1)$ to be a uniform density, and we note that $p(X | w_B, w_C)$ is simply the exponential of $\mathcal{L}(X | w_B, w_C)$ as defined in Equation 5. $P(X | h_C = 0)$, the marginal likelihood for $\mathrm{Graph}_0$, is computed similarly, but with the prior $p(w_B, w_C | h_C = 1)$ in Equation 12 replaced by $p(w_B | h_C = 0)\delta(w_C)$. We again take $p(w_B | h_C = 0)$ to be uniform. The Dirac delta distribution on $w_C = 0$ enforces the restriction that the $C \rightarrow E$ link is absent. By making these assumptions, we eliminate the need for any free numerical parameters in our probabilistic model (in contrast to a similar Bayesian account proposed by Anderson [1]).

Because causal support depends on the full likelihood functions for both $\mathrm{Graph}_1$ and $\mathrm{Graph}_0$, we may expect the support model to be modulated by causal power – which is based strictly on the likelihood maximum estimate for $\mathrm{Graph}_1$ – but only in interaction with other factors that determine how much of the posterior probability mass for $w_C$ in $\mathrm{Graph}_1$ is bounded away from zero (where it is pinned in $\mathrm{Graph}_0$). In general, evaluating causal support may require fairly involved computations, but in the limit of large $N$ and weak causal strength $w_C$, it can be approximated by the familiar $\chi^2$ statistic for independence, $N \sum_{c,e} \frac{(P(c,e) - P_0(c,e))^2}{P_0(c,e)}$. Here $P_0(c,e) = P(c)P(e)$ is the factorized approximation to $P(c,e)$, which assumes $C$ and $E$ to be independent (as they are in $\mathrm{Graph}_0$).

## 3 Comparison with experiments

In this section we examine the strengths and weaknesses of the two parameter inference models, $\Delta P$ and causal power, and the two structural inference models, causal support and $\chi^2$, as accounts of data from three behavioral experiments, each designed to address different aspects of human causal induction. To compensate for possible nonlinearities in people's use of numerical rating scales on these tasks, all model predictions have been scaled by power-law transformations, $f(x) = \text{sign}(x)|x|^\gamma$, with $\gamma$ chosen separately for each model

and each data set to maximize their linear correlation. In the figures, predictions are expressed over the same range as the data, with minimum and maximum values aligned.

Figure 2 presents data from a study by Buehner & Cheng [2], designed to contrast the predictions of $\Delta P$ and causal power. People judged $P(C{\rightarrow}E)$ for hypothetical medical studies much like the gene expression scenarios described above, seeing eight cases in which $C$ occurred and eight in which $C$ did not occur. Some trends in the data are clearly captured by the causal power model but not by $\Delta P$, such as the monotonic decrease in $P(C{\rightarrow}E)$ from $\{1.00, 0.75\}$ to $\{.25, 0.00\}$, as $\Delta P$ stays constant but $P(e^+|c^-)$ (and hence power) decreases (columns 6-9). Other trends are clearly captured by $\Delta P$ but not by the power model, like the monotonic increase in $P(C{\rightarrow}E)$ as $P(e^+|c^+)$ stays constant at 1.0 but $P(e^+|c^-)$ decreases, from $\{1.00, 1.00\}$ to $\{1.00, 0.00\}$ (columns 1, 6, 10, 13, 15). However, one of the most salient trends is captured by neither model: the decrease in $P(C{\rightarrow}E)$ as $\Delta P$ stays constant at 0 but $P(e^+|c^-)$ decreases (columns 1-5). The causal support model predicts this decrease, as well as the other trends. The intuition behind the model's predictions for $\Delta P = 0$ is that decreasing the base rate $P(e^+|c^-)$ increases the opportunity to observe the cause's influence and thus increases the statistical force behind the inference that $C$ does not cause $E$, given $\Delta P = 0$. This effect is most obvious when $P(e^+|c^+) = P(e^+|c^-) = 1$, yielding a ceiling effect with no statistical leverage [3], but also occurs to a lesser extent for $P(e^+|c^-) < 1$. While $\chi^2$ generally approximates the support model rather well, it also fails to explain the cases with $P(e^+|c^+) = P(e^+|c^-)$, which always yield $\chi^2 = 0$. The superior fit of the support model is reflected in its correlation with the data, giving $R^2 = 0.95$ while the power, $\Delta P$, and $\chi^2$ models gave $R^2$ values of $0.81, 0.82$, and $0.82$ respectively.

Figure 3 shows results from an experiment conducted by Lober and Shanks [6], designed to explore the trend in Buehner and Cheng's experiment that was predicted by $\Delta P$ but not by the power model. Columns 4-7 replicated the monotonic increase in $P(C{\rightarrow}E)$ when $P(e^+|c^+)$ remains constant at 1.0 but $P(e^+|c^-)$ decreases, this time with 28 cases in which $C$ occurred and 28 in which $C$ did not occur. Columns 1-3 show a second situation in which the predictions of the power model are constant, but judgements of $P(C{\rightarrow}E)$ increase. Columns 8-10 feature three scenarios with equal $\Delta P$, for which the causal power model predicts a decreasing trend. These effects were explored by presenting a total of 60 trials, rather than the 56 used in Columns 4-7. For each of these trends the $\Delta P$ model outperforms the causal power model, with overall $R^2$ values of 0.96 and 0.36 respectively. However, it is important to note that the responses of the human subjects in columns 8-10 (contingencies $\{1.00, 0.60\}, \{0.80, 0.40\}, \{0.40, 0.00\}$) are not quite consistent with the predictions of $\Delta P$: they show a slight U-shaped non-linearity, with $P(C{\rightarrow}E)$ judged to be smaller for $0.80, 0.40$ than for either of the extreme cases. This trend is predicted by the causal support model and its $\chi^2$ approximation, however, which both give the slightly better $R^2$ of 0.99.

Figure 4 shows data that we collected in a similar survey, aiming to explore this non-linear effect in greater depth. 35 students in an introductory psychology class completed the survey for partial course credit. They each provided a judgment of $P(C{\rightarrow}E)$ in 14 different medical scenarios, where information about $P(e^+|c^+)$ and $P(e^+|c^-)$ was provided in terms of how many mice from a sample of 100 expressed a particular gene. Columns 1-3, 5-7, and 9-11 show contingency structures designed to elicit U-shaped trends in $P(C{\rightarrow}E)$. Columns 4 and 8 give intermediate values, also consistent with the observed non-linearity. Column 14 attempted to explore the effects of manipulating sample size, with a contingency structure of $\{7/7, 93/193\}$. In each case, we observed the predicted nonlinearity: in a set of situations with the same $\Delta P$, the situations involving less extreme probabilities show reduced judgments of $P(C{\rightarrow}E)$. These non-linearities are not consistent with the $\Delta P$ model, but

are predicted by both causal support and $\chi^2$. $\Delta P$ actually achieves a correlation comparable to $\chi^2$ ($R^2 = 0.92$ for both models) because the non-linear effects contribute only weakly to the total variance. The support model gives a slightly worse fit than $\chi^2$, $R^2 = 0.80$, while the power model gives a poor account of the data, $R^2 = 0.38$.

## 4   Conclusions and future directions

In each of the studies above, the structural inference models based on causal support or $\chi^2$ consistently outperformed the parameter estimation models, $\Delta P$ and causal power. While causal power and $\Delta P$ were each capable of capturing certain trends in the data, causal support was the only model capable of predicting all the trends. For the third data set, $\chi^2$ provided a significantly better fit to the data than did causal support. This finding merits future investigation in a study designed to tease apart $\chi^2$ and causal support; in any case, due to the close relationship between the two models, this result does not undermine our claim that probabilistic structural inferences are central to human causal induction.

One unique advantage of the Bayesian causal support model is its ability to draw inferences from very few observations. We have begun a line of experiments, inspired by Gopnik, Sobel & Glymour (submitted), to examine how adults revise their causal judgments when given only one or two observations, rather than the large samples used in the above studies. In one study, subjects were faced with a machine that would inform them whether a pencil placed upon it contained "superlead" or ordinary lead. Subjects were either given prior knowledge that superlead was rare or that it was common. They were then given two pencils, analogous to $B$ and $C$ in Figure 1, and asked to rate how likely these pencils were to have superlead, that is, to cause the detector to activate. Mean responses reflected the induced prior. Next, they were shown that the superlead detector responded when $B$ and $C$ were tested together, and their causal ratings of both $B$ and $C$ increased. Finally, they were shown that $B$ set off the superlead detector on its own, and causal ratings of $B$ increased to ceiling while ratings of $C$ returned to their prior levels. This situation is exactly analogous to that explored in the medical tasks described above, and people were able to perform accurate causal inductions given only one trial of each type. Of the models we have considered, only Bayesian causal support can explain this behavior, by allowing the prior in Equation 11 to adapt depending on whether superlead is rare or common.

We also hope to look at inferences about more complex causal structures, including those with hidden variables. With just a single cause, causal support and $\chi^2$ are highly correlated, but with more complex structures, the Bayesian computation of causal support becomes increasingly intractable while the $\chi^2$ approximation becomes less accurate. Through experiments with more complex structures, we hope to discover where and how human causal induction strikes a balance between ponderous rationality and efficient heuristic.

Finally, we should stress that despite the superior performance of the structural inference models here, in many situations estimating causal strength parameters is likely to be just as important as inferring causal structure. Our hope is that by using graphical models to relate and extend upon existing accounts of causal induction, we have provided a framework for exploring the interplay between the different kinds of judgments that people make.

### References

[1] J. Anderson (1990). *The adaptive character of thought*. Erlbaum.
[2] M. Buehner & P. Cheng (1997) Causal induction; The power PC theory versus the Rescorla-Wagner theory. In *Proceedings of the 19th Annual Conference of the Cognitive Science Society*.

[3] P. Cheng (1997). From covariation to causation: A causal power theory. *Psychological Review* **104**, 367-405.

[4] T. Cover & J .Thomas (1991). *Elements of information theory*. Wiley.

[5] C. Glymour (1998). Learning causes: Psychological explanations of causal explanation. *Minds and Machines* **8**, 39-60.

[6] K. Lober & D. Shanks (2000). Is causal induction based on causal power? Critique of Cheng (1997). *Psychological Review* **107**, 195-212.

[7] J. Pearl (2000). *Causality*. Cambridge University Press.

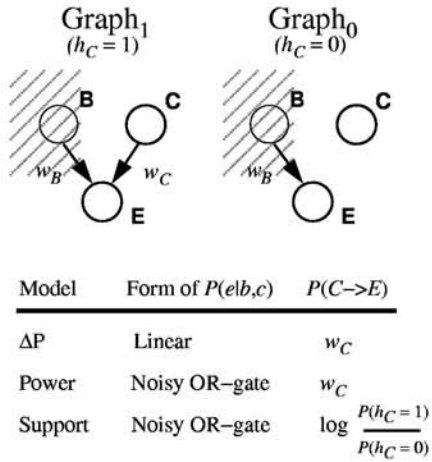

| Model | Form of $P(e|b,c)$ | $P(C{\rightarrow}E)$ |
|---|---|---|
| $\Delta P$ | Linear | $w_C$ |
| Power | Noisy OR–gate | $w_C$ |
| Support | Noisy OR–gate | $\log \dfrac{P(h_C=1)}{P(h_C=0)}$ |

**Figure 1:** Different theories of human causal induction expressed as different operations on a simple graphical model. The $\Delta P$ and power models correspond to maximum likelihood parameter estimates on a fixed graph (Graph$_1$), while the support model corresponds to a (Bayesian) inference about which graph is the true causal structure.

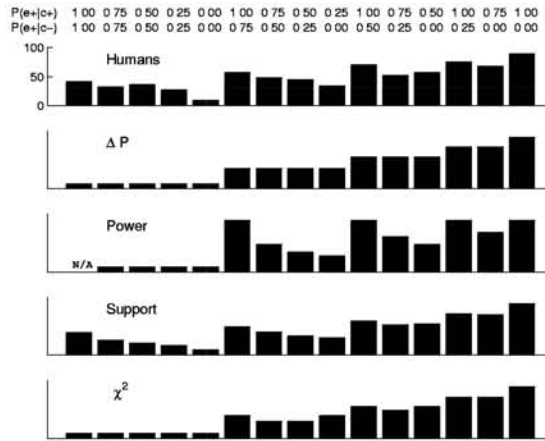

**Figure 2:** Computational models compared with the performance of human participants from Buehner and Cheng [1], Experiment 1B. Numbers along the top of the figure show stimulus contingencies.

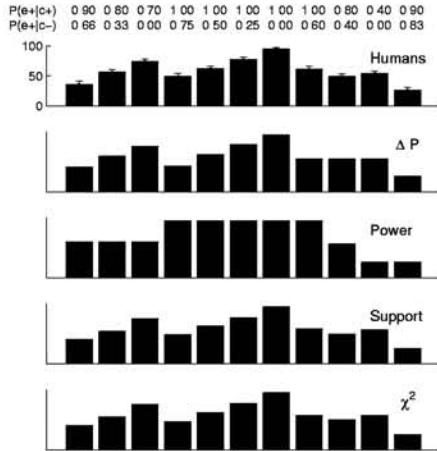

**Figure 3:** Computational models compared with the performance of human participants from Lober and Shanks [5], Experiments 4–6.

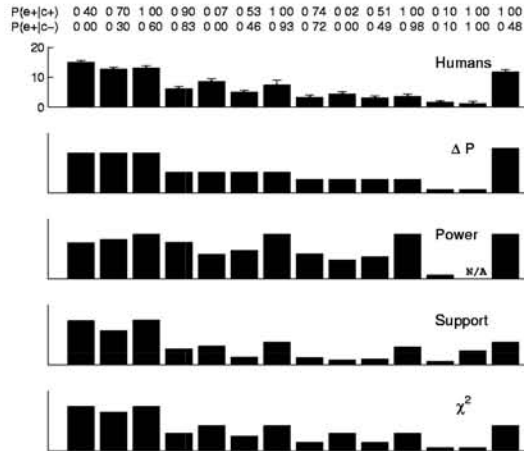

**Figure 4:** Computational models compared with the performance of human participants on a set of stimuli designed to elicit the non–monotonic trends shown in the data of Lober and Shanks [5].